# Abstraction and relational learning

**Charles Kemp & Alan Jern**
Department of Psychology
Carnegie Mellon University
{ckemp,ajern}@cmu.edu

## Abstract

Most models of categorization learn categories defined by characteristic features but some categories are described more naturally in terms of relations. We present a generative model that helps to explain how relational categories are learned and used. Our model learns abstract schemata that specify the relational similarities shared by instances of a category, and our emphasis on abstraction departs from previous theoretical proposals that focus instead on comparison of concrete instances. Our first experiment suggests that abstraction can help to explain some of the findings that have previously been used to support comparison-based approaches. Our second experiment focuses on one-shot schema learning, a problem that raises challenges for comparison-based approaches but is handled naturally by our abstraction-based account.

Categories such as *family*, *sonnet*, *above*, *betray*, and *imitate* differ in many respects but all of them depend critically on relational information. Members of a family are typically related by blood or marriage, and the lines that make up a sonnet must rhyme with each other according to a certain pattern. A pair of objects will demonstrate "aboveness" only if a certain spatial relationship is present, and an event will qualify as an instance of betrayal or imitation only if its participants relate to each other in certain ways. All of the cases just described are examples of relational categories. This paper develops a computational approach that helps to explain how simple relational categories are acquired.

Our approach highlights the role of abstraction in relational learning. Given several instances of a relational category, it is often possible to infer an abstract representation that captures what the instances have in common. We refer to these abstract representations as *schemata*, although others may prefer to call them rules or theories. For example, a sonnet schema might specify the number of lines that a sonnet should include and the rhyming pattern that the lines should follow. Once a schema has been acquired it can support several kinds of inferences. A schema can be used to make predictions about hidden aspects of the examples already observed—if the final word in a sonnet is illegible, the rhyming pattern can help to predict the identity of this word. A schema can be used to decide whether new examples (e.g. new poems) qualify as members of the category. Finally, a schema can be used to generate novel examples of a category (e.g. novel sonnets).

Most researchers would agree that abstraction plays some role in relational learning, but Gentner [1] and other psychologists have emphasized the role of *comparison* instead [2, 3]. Given one example of a sonnet and the task of deciding whether a second poem is also a sonnet, a comparison-based approach might attempt to establish an alignment or mapping between the two. Approaches that rely on comparison or mapping are especially prominent in the literature on analogical reasoning [4, 5], and many of these approaches can be viewed as accounts of relational categorization [6]. For example, the problem of deciding whether two systems are analogous can be formalized as the problem of deciding whether these systems are instances of the same relational category. Despite some notable exceptions [6, 7], most accounts of analogy focus on comparison rather than abstraction, and suggest that "analogy passes from one instance of a generalization to another without pausing for explicit induction of the generalization" (p 95) [8].

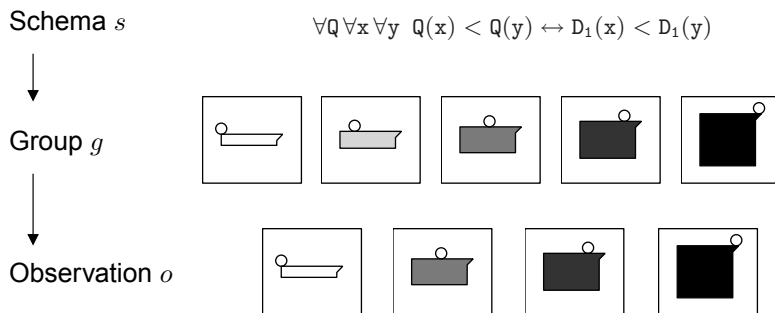

Figure 1: A hierarchical generative model for learning and using relational categories. The schema $s$ at the top level is a logical sentence that specifies which groups are valid instances of the category. The group $g$ at the second level is randomly sampled from the set of valid instances, and the observation $o$ is a partially observed version of group $g$.

Researchers that focus on comparison sometimes discuss abstraction, but typically suggest that abstractions emerge as a consequence of comparing two or more concrete instances of a category [3, 5, 9, 10]. This view, however, will not account for one-shot inferences, or inferences based on a single instance of a relational category. Consider a learner who is shown one instance of a sonnet then asked to create a second instance. Since only one instance is provided, it is hard to see how comparisons between instances could account for success on the task. A single instance, however, will sometimes provide enough information for a schema to be learned, and this schema should allow subsequent instances to be generated [11]. Here we develop a formal framework for exploring relational learning in general and one-shot schema learning in particular.

Our framework relies on the hierarchical Bayesian approach, which provides a natural way to combine abstraction and probabilistic inference [12]. The hierarchical Bayesian approach supports representations at multiple levels of abstraction, and helps to explains how abstract representations (e.g. a sonnet schema) can be acquired given observations of concrete instances (e.g. individual sonnets). The schemata we consider are represented as sentences in a logical language, and our approach therefore builds on previous probabilistic methods for learning and using logical theories [13, 14]. Following previous authors, we propose that logical representations can help to capture the content of human knowledge, and that Bayesian inference helps to explain how these representations are acquired and how they support inductive inference.

The following sections introduce our framework then evaluate it using two behavioral experiments. Our first experiment uses a standard classification task where participants are shown one example of a category then asked to decide which of two alternatives is more likely to belong to the same category. Tasks of this kind have previously been used to argue for the importance of comparison, but we suggest that these tasks can be handled by accounts that focus on abstraction. Our second experiment uses a less standard generation task [15, 16] where participants are shown a single example of a category then asked to generate additional examples. As predicted by our abstraction-based account, we find that people are able to learn relational categories on the basis of a single example.

## 1 A generative approach to relational learning

Our examples so far have used real-world relational categories such as *family* and *sonnet* but we now turn to a very simple domain where relational categorization can be studied. Each element in the domain is a group of components that vary along a number of dimensions—in Figure 1, the components are figures that vary along the dimensions of size, color, and circle position. The groups can be organized into categories—one such category includes groups where every component is black. Although our domain is rather basic it allows some simple relational regularities to be explored. We can consider categories, for example, where all components in a group must be the *same* along some dimension, and categories where all components must be *different* along some dimension. We can also consider categories defined by relationships between dimensions—for example, the category that includes all groups where the size and color dimensions are correlated.

Each category is associated with a schema, or an abstract representation that specifies which groups are valid instances of the category. Here we consider schemata that correspond to rules formulated

$$1 \quad \begin{Bmatrix} \forall x \\ \exists x \end{Bmatrix} D_i(x) \{=,\neq,<,>\} v_k$$

$$2 \quad \begin{Bmatrix} \forall x \\ \exists x \end{Bmatrix} \begin{Bmatrix} \forall y\ x \neq y \rightarrow \\ \exists y\ x \neq y \wedge \end{Bmatrix} D_i(x) \{=,\neq,<,>\} D_i(y)$$

$$3 \quad \forall x\ D_i(x) \{=,\neq\} v_k \begin{Bmatrix} \wedge \\ \vee \\ \leftrightarrow \end{Bmatrix} D_j(x) \{=,\neq\} v_l$$

$$4 \quad \forall x \forall y\ x \neq y \rightarrow \left( D_i(x) \{=,\neq,<,>\} D_i(y) \begin{Bmatrix} \wedge \\ \vee \\ \leftrightarrow \end{Bmatrix} D_j(x) \{=,\neq,<,>\} D_j(y) \right)$$

$$5 \quad \begin{Bmatrix} \forall Q \\ \exists Q \end{Bmatrix} \begin{Bmatrix} \forall x \\ \exists x \end{Bmatrix} \begin{Bmatrix} \forall y\ x \neq y \rightarrow \\ \exists y\ x \neq y \wedge \end{Bmatrix} Q(x) \{=,\neq,<,>\} Q(y)$$

$$6 \quad \begin{Bmatrix} \forall Q\ Q \neq D_i \rightarrow \\ \exists Q\ Q \neq D_i \wedge \end{Bmatrix} \forall x \forall y\ x \neq y \rightarrow \left( Q(x) \{=,\neq,<,>\} Q(y) \begin{Bmatrix} \wedge \\ \vee \\ \leftrightarrow \end{Bmatrix} D_i(x) \{=,\neq,<,>\} D_i(y) \right)$$

$$7 \quad \begin{Bmatrix} \forall Q \\ \exists Q \end{Bmatrix} \begin{Bmatrix} \forall R\ Q \neq R \rightarrow \\ \exists R\ Q \neq R \wedge \end{Bmatrix} \forall x \forall y\ x \neq y \rightarrow \left( Q(x) \{=,\neq,<,>\} Q(y) \begin{Bmatrix} \wedge \\ \vee \\ \leftrightarrow \end{Bmatrix} R(x) \{=,\neq,<,>\} R(y) \right)$$

Table 1: Templates used to construct a hypothesis space of logical schemata. An instance of a given template can be created by choosing an element from each set enclosed in braces (some sets are laid out horizontally to save space), replacing each occurrence of $D_i$ or $D_j$ with a dimension (e.g. $D_1$) and replacing each occurrence of $v_k$ or $v_l$ with a value (e.g. 1).

in a logical language. The language includes three binary connectives—and ($\wedge$), or ($\vee$), and if and only if ($\leftrightarrow$). Four binary relations ($=$, $\neq$, $<$, and $>$) are available for comparing values along dimensions. Universal quantification ($\forall x$) and existential quantification ($\exists x$) are both permitted, and the language includes quantification over objects ($\forall x$) and dimensions ($\forall Q$). For example, the schema in Figure 1 states that all dimensions are aligned. More precisely, if $D_1$ is the dimension of size, the schema states that for all dimensions $Q$, a component $x$ is smaller than a component $y$ along dimension $Q$ if and only if $x$ is smaller in size than $y$. It follows that all three dimensions must increase or decrease together.

To explain how rules in this logical language are learned we work with the hierarchical generative model in Figure 1. The representation at the top level is a schema $s$, and we assume that one or more groups $g$ are generated from a distribution $P(g|s)$. Following a standard approach to category learning [17, 18], we assume that $g$ is uniformly sampled from all groups consistent with $s$:

$$p(g|s) \propto \begin{cases} 1 & g \text{ is consistent with } s \\ 0 & \text{otherwise} \end{cases} \qquad (1)$$

For all applications in this paper, we assume that the number of components in a group is known and fixed in advance.

The bottom level of the hierarchy specifies observations $o$ that are generated from a distribution $P(o|g)$. In most cases we assume that $g$ can be directly observed, and that $P(o|g) = 1$ if $o = g$ and 0 otherwise. We also consider the setting shown in Figure 1 where $o$ is generated by concealing a component of $g$ chosen uniformly at random. Note that the observation $o$ in Figure 1 includes only four of the components in group $g$, and is roughly analogous to our earlier example of a sonnet with an illegible final word.

To convert Figure 1 into a fully-specified probabilistic model it remains to define a prior distribution $P(s)$ over schemata. An appealing approach is to consider all of the infinitely many sentences in the logical language already mentioned, and to define a prior favoring schemata which correspond to simple (i.e. short) sentences. We approximate this approach by considering a large but finite space of sentences that includes all instances of the templates in Table 1 and all conjunctions of these instances. When instantiating one of these templates, each occurrence of $D_i$ or $D_j$ should be replaced by one of the dimensions in the domain. For example, the schema in Figure 1 is a simplified instance of template 6 where $D_i$ is replaced by $D_1$. Similarly, each instance of $v_k$ or $v_l$ should be replaced by a value along one of the dimensions. Our first experiment considers a problem where there are are three dimensions and three possible values along each dimension (i.e. $v_k = 1, 2,$ or 3). As a result there are 1568 distinct instances of the templates in Table 1 and roughly one million

conjunctions of these instances. Our second experiment uses three dimensions with five values along each dimension, which leads to 2768 template instances and roughly three million conjunctions of these instances.

The templates in Table 1 capture most of the simple regularities that can be formulated in our logical language. Template 1 generates all rules that include quantification over a single object variable and no binary connectives. Template 3 is similar but includes a single binary connective. Templates 2 and 4 are similar to 1 and 3 respectively, but include two object variables ($x$ and $y$) rather than one. Templates 5, 6 and 7 add quantification over dimensions to Templates 2 and 4. Although the templates in Table 1 capture a large class of regularities, several kinds of templates are not included. Since we do not assume that the dimensions are commensurable, values along different dimensions cannot be directly compared ($\exists x\, D_1(x) = D_2(x)$ is not permitted. For the same reason, comparisons to a dimension value must involve a concrete dimension ($\forall x\, D_1(x) = 1$ is permitted) rather than a dimension variable ($\forall Q\, \forall x\, Q(x) = 1$ is not permitted). Finally, we exclude all schemata where quantification over objects precedes quantification over dimensions, and as a result there are some simple schemata that our implementation cannot learn (e.g. $\exists x \forall y \exists Q\, Q(x) = Q(y)$).

The extension of each schema is a set of groups, and schemata with the same extension can be assigned to the same equivalence class. For example, $\forall x\, D_1(x) = v_1$ (an instance of template 1) and $\forall x\, D_1(x) = v_1 \wedge D_1(x) = v_1$ (an instance of template 3) end up in the same equivalence class. Each equivalence class can be represented by the shortest sentence that it contains, and we define our prior $P(s)$ over a set that includes a single representative for each equivalence class. The prior probability $P(s)$ of each sentence is inversely proportional to its length: $P(s) \propto \lambda^{|s|}$, where $|s|$ is the length of schema $s$ and $\lambda$ is a constant between 0 and 1. For all applications in this paper we set $\lambda = 0.8$.

The generative model in Figure 1 can be used for several purposes, including schema learning (inferring a schema $s$ given one or more instances generated from the schema), classification (deciding whether group $g_{new}$ belongs to a category given one or more instances of the category) and generation (generating a group $g_{new}$ that belongs to the same category as one or more instances). Our first experiment explores all three of these problems.

## 2   Experiment 1: Relational classification

Our first experiment is organized around a triad task where participants are shown one example of a category then asked to decide which of two choice examples is more likely to belong to the category. Triad tasks are regularly used by studies of relational categorization, and have been used to argue for the importance of comparison [1]. A comparison-based approach to this task, for instance, might compare the example object to each of the choice objects in order to decide which is the better match. Our first experiment is intended in part to explore whether a schema-learning approach can also account for inferences about triad tasks.

**Materials and Method.** 18 adults participated for course credit and interacted with a custom-built computer interface. The stimuli were groups of figures that varied along three dimensions (color, size, and ball position, as in Figure 1). Each shape was displayed on a single card, and all groups in Experiment 1 included exactly three cards. The cards in Figure 1 show five different values along each dimension, but Experiment 1 used only three values along each dimension.

The experiment included inferences about 10 triads. Participants were told that aliens from a certain planet "enjoy organizing cards into groups," and that "any group of cards will probably be liked by some aliens and disliked by others." The ten triad tasks were framed as questions about the preferences of 10 aliens. Participants were shown a group that Mr X likes (different names were used for the ten triads), then shown two choice groups and told that "Mr X likes one of these groups but not the other." Participants were asked to select one of the choice groups, then asked to generate another 3-card group that Mr X would probably like. Cards could be added to the screen using an "Add Card" button, and there were three pairs of buttons that allowed each card to be increased or decreased along the three dimensions. Finally, participants were asked to explain in writing "what kind of groups Mr X likes."

The ten triads used are shown in Figure 2. Each group is represented as a 3 by 3 matrix where rows represent cards and columns show values along the three dimensions. Triad 1, for example,

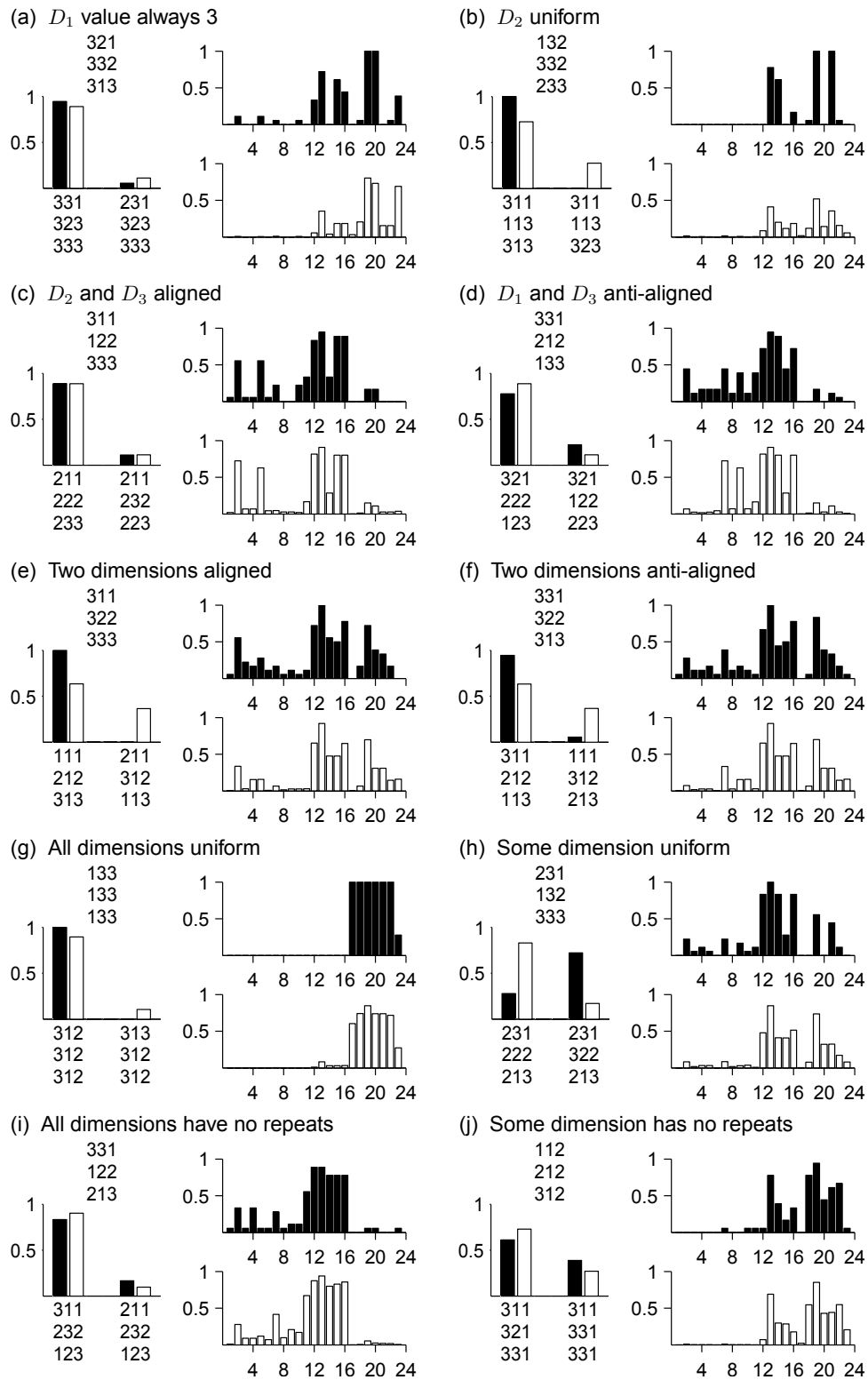

Figure 2: Human responses and model predictions for the ten triads in Experiment 1. The plot at the left of each panel shows model predictions (white bars) and human preferences (black bars) for the two choice groups in each triad. The plots at the right of each panel summarize the groups created during the generation phase. The 23 elements along the x-axis correspond to the regularities listed in Table 2.

| 1 | All dimensions aligned | 13 | One dimension has no repeats |
|---|---|---|---|
| 2 | Two dimensions aligned | 14 | $D_1$ has no repeats |
| 3 | $D_1$ and $D_2$ aligned | 15 | $D_2$ has no repeats |
| 4 | $D_1$ and $D_3$ aligned | 16 | $D_3$ has no repeats |
| 5 | $D_2$ and $D_3$ aligned | 17 | All dimensions uniform |
| 6 | All dimensions aligned or anti-aligned | 18 | Two dimensions uniform |
| 7 | Two dimensions anti-aligned | 19 | One dimension uniform |
| 8 | $D_1$ and $D_2$ anti-aligned | 20 | $D_1$ uniform |
| 9 | $D_1$ and $D_3$ anti-aligned | 21 | $D_2$ uniform |
| 10 | $D_2$ and $D_3$ anti-aligned | 22 | $D_3$ uniform |
| 11 | All dimensions have no repeats | 23 | $D_1$ value is always 3 |
| 12 | Two dimensions have no repeats | | |

Table 2: Regularities used to code responses to the generation tasks in Experiments 1 and 2

has an example group including three cards that each take value 3 along $D_1$. The first choice group is consistent with this regularity but the second choice group is not. The cards in each group were arrayed vertically on screen, and were initially sorted as shown in Figure 2 (i.e. first by $D_3$, then by $D_2$ and then by $D_1$). The cards could be dragged around on screen, and participants were invited to move them around in order to help them understand each group. The mapping between the three dimensions in each matrix and the three dimensions in the experiment (color, position, and size) was randomized across participants, and the order in which triads were presented was also randomized.

**Model predictions and results.** Let $g_e$ be the example group presented in the triad task and $g_1$ and $g_2$ be the two choice groups. We use our model to compute the relative probability of two hypotheses: $h_1$ which states that $g_e$ and $g_1$ are generated from the same schema and that $g_2$ is sampled randomly from all possible groups, and $h_2$ which states that $g_e$ and $g_2$ are generated from the same schema. We set $P(h_1) = P(h_2) = 0.5$, and compute posterior probabilities $P(h_1|g_e, g_1, g_2)$ and $P(h_2|g_e, g_1, g_2)$ by integrating over all schemata in the hypothesis space already described. Our model assumes that two groups are considered similar to the extent that they appear to have been generated by the same underlying schema, and is consistent with the generative approach to similarity described by Kemp et al. [19].

Model predictions for the ten triads are shown in Figure 2. In each case, the choice probabilities plotted (white bars) are the posterior probabilities of hypotheses $h_1$ and $h_2$. In nine out of ten cases the best choice according to the model is the most common human response. Responses to triads 2c and 2d support the idea that people are sensitive to relationships between dimensions (i.e. alignment and anti-alignment). Triads 2e and 2f are similar to triads studied by Kotovsky and Gentner [1], and we replicate their finding that people are sensitive to relationships between dimensions even when the dimensions involved vary from group to group. The one case where human responses diverge from model predictions is shown in Figure 2h. Note that the schema for this triad involves existential quantification over dimensions (*some* dimension is uniform), and according to our prior $P(s)$ this kind of quantification is no more complex than other kinds of quantification. Future applications of our approach can explore the idea that existential quantification over dimensions ($\exists Q$) is psychologically more complex than universal quantification over dimensions ($\forall Q$) or existential quantification over cards ($\exists x$), and can consider logical languages that incorporate this inductive bias.

To model the generation phase of the experiment we computed the posterior distribution

$$P(g_{\text{new}}|g_e, g_1, g_2) = \sum_{s,h} P(g_{\text{new}}|s) P(s|h, g_e, g_1, g_2) P(h|g_e, g_1, g_2)$$

where $P(h|g_e, g_1, g_2)$ is the distribution used to model selections in the triad task. Since the space of possible groups is large, we visualize this distribution using a profile that shows the posterior probability assigned to groups consistent with the 23 regularities shown in Table 2. The white bar plots in Figure 2 show profiles predicted by the model, and the black plots immediately above show profiles computed over the groups generated by our 18 participants.

In many of the 10 cases the model accurately predicts regularities in the groups generated by people. In case 2c, for example, the model correctly predicts that generated groups will tend to have no repeats along dimensions $D_2$ and $D_3$ (regularities 15 and 16) and that these two dimensions will be aligned (regularities 2 and 5). There are, however, some departures from the model's predictions, and a notable example occurs in case 2d. Here the model detects the regularity that dimensions $D_1$ and $D_3$ are anti-aligned (regularity 9). Some groups generated by participants are consistent with

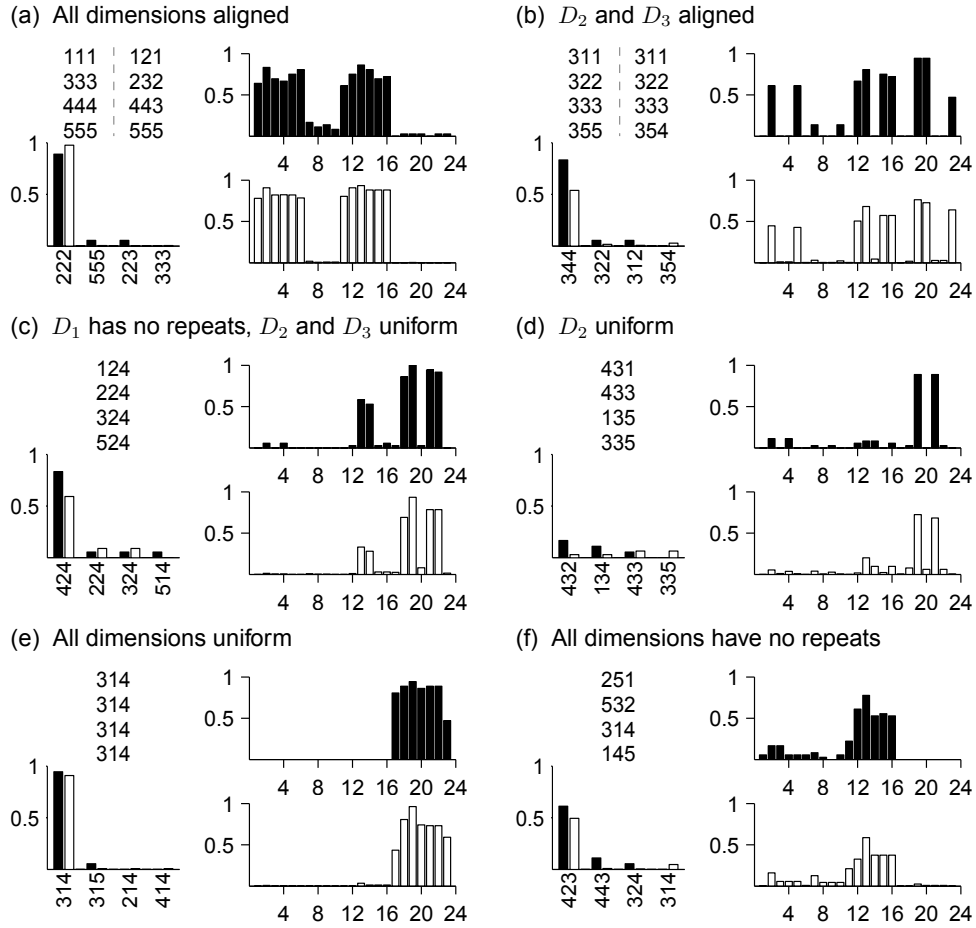

Figure 3: Human responses and model predictions for the six cases in Experiment 2. In (a) and (b), the 4 cards used for the completion and generation phases are shown on either side of the dashed line (completion cards on the left). In the remaining cases, the same 4 cards were used for both phases. The plots at the right of each panel show model predictions (white bars) and human responses (black bars) for the generation task. In each case, the 23 elements along each x-axis correspond to the regularities listed in Table 2. The remaining plots show responses to the completion task. There are 125 possible responses, and the four responses shown always include the top two human responses and the top two model predictions.

this regularity, but people also regularly generate groups where two dimensions are aligned rather than anti-aligned (regularity 2). This result may indicate that some participants are sensitive to relationships between dimensions but do not consider the difference between a positive relationship (alignment) and an inverse relationship (anti-alignment) especially important.

Kotovsky and Gentner [1] suggest that comparison can explain how people respond to triad tasks, although they do not provide a computational model that can be compared with our approach. It is less clear how comparison might account for our generation data, and our next experiment considers a one-shot generation task that raises even greater challenges for a comparison-based approach.

## 3 Experiment 2: One-shot schema learning

As described already, comparison involves constructing mappings between pairs of category instances. In some settings, however, learners make confident inferences given a single instance of a category [15, 20], and it is difficult to see how comparison could play a major role when only one instance is available. Models that rely on abstraction, however, can naturally account for one-shot relational learning, and we designed a second experiment to evaluate this aspect of our approach.

Several previous studies have explored one-shot relational learning. Holyoak and Thagard [21] developed a study of analogical reasoning using stories as stimuli and found little evidence of one-shot schema learning. Ahn et al. [11] demonstrated, however, that one-shot learning can be achieved with complex materials such as stories, and modeled this result using explanation-based learning. Here we use much simpler stimuli and explore a probabilistic approach to one-shot learning.

**Materials and Method.** 18 adults participated for course credit. The same individuals completed Experiments 1 and 2, and Experiment 2 was always run before Experiment 1. The same computer interface was used in both experiments, and the only important difference was that the figures in Experiment 2 could now take five values along each dimension rather than three.

The experiment included two phases. During the generation phase, participants saw a 4-card group that Mr X liked and were asked to generate two 5-card groups that Mr X would probably like. During the completion phase, participants were shown four members of a 5-card group and were asked to generate the missing card. The stimuli used in each phase are shown in Figure 3. In the first two cases, slightly different stimuli were used in the generation and completion phases, and in all remaining cases the same set of four cards was used in both cases. All participants responded to the six generation questions before answering the six completion questions.

**Model predictions and results.** The generation phase is modeled as in Experiment 1, but now the posterior distribution $P(g_{new}|g_e)$ is computed after observing a single instance of a category. The human responses in Figure 3 (white bars) are consistent with the model in all cases, and confirm that a single example can provide sufficient evidence for learners to acquire a relational category. For example, the most common response in case 3a was the 5-card group shown in Figure 1—a group with all three dimensions aligned.

To model the completion phase, let $o_e$ represent a partial observation of group $g_e$. Our model infers which card is missing from $g_e$ by computing the posterior distribution $P(g_e|o_e) \propto P(o_e|g_e) \sum_s P(g_e|s)P(s)$, where $P(o_e|g_e)$ captures the idea that $o_e$ is generated by randomly concealing one component of $g_e$. The white bars in Figure 3 show model predictions, and in five out of six cases the best response according to the model is the same as the most common human response. In the remaining case (Figure 3d) the model generates a diffuse distribution over all cards with value 3 on dimension 2, and all human responses satisfy this regularity.

## 4 Conclusion

We presented a generative model that helps to explain how relational categories are learned and used. Our approach captures relational regularities using a logical language, and helps to explain how schemata formulated in this language can be learned from observed data. Our approach differs in several respects from previous accounts of relational categorization [1, 5, 10, 22]. First, we focus on abstraction rather than comparison. Second, we consider tasks where participants must generate examples of categories [16] rather than simply classify existing examples. Finally, we provide a formal account that helps to explain how relational categories can be learned from a single instance.

Our approach can be developed and extended in several ways. For simplicity, we implemented our model by working with a finite space of several million schemata, but future work can consider hypothesis spaces that assign non-zero probability to all regularities that can be formulated in the language we described. The specific logical language used here is only a starting point, and future work can aim to develop languages that provide a more faithful account of human inductive biases. Finally, we worked with a domain that provides one of the simplest ways to address core questions such as one-shot learning. Future applications of our general approach can consider domains that include more than three dimensions and a richer space of relational regularities.

Relational learning and analogical reasoning are tightly linked, and hierarchical generative models provide a promising approach to both problems. We focused here on relational categorization, but future studies can explore whether probabilistic accounts of schema learning can help to explain the inductive inferences typically considered by studies of analogical reasoning. Although there are many models of analogical reasoning, there are few that pursue a principled probabilistic approach, and the hierarchical Bayesian approach may help to fill this gap in the literature.

**Acknowledgments** We thank Maureen Satyshur for running the experiments. This work was supported in part by NSF grant CDI-0835797.

# References

[1] L. Kotovsky and D. Gentner. Comparison and categorization in the development of relational similarity. *Child Development*, 67:2797–2822, 1996.

[2] D. Gentner and A. B. Markman. Structure mapping in analogy and similarity. *American Psychologist*, 52:45–56, 1997.

[3] D. Gentner and J. Medina. Similarity and the development of rules. *Cognition*, 65:263–297, 1998.

[4] B. Falkenhainer, K. D. Forbus, and D. Gentner. The structure-mapping engine: Algorithm and examples. *Artificial Intelligence*, 41:1–63, 1989.

[5] J. E. Hummel and K. J. Holyoak. A symbolic-connectionist theory of relational inference and generalization. *Psychological Review*, 110:220–264, 2003.

[6] M. Mitchell. *Analogy-making as perception: a computer model*. MIT Press, Cambridge, MA, 1993.

[7] D. R. Hofstadter and the Fluid Analogies Research Group. *Fluid concepts and creative analogies: computer models of the fundamental mechanisms of thought*. 1995.

[8] W. V. O. Quine and J. Ullian. *The Web of Belief*. Random House, New York, 1978.

[9] J. Skorstad, D. Gentner, and D. Medin. Abstraction processes during concept learning: a structural view. In *Proceedings of the 10th Annual Conference of the Cognitive Science Society*, pages 419–425. 2009.

[10] D. Gentner and J. Loewenstein. Relational language and relational thought. In E. Amsel and J. P. Byrnes, editors, *Language, literacy and cognitive development: the development and consequences of symbolic communication*, pages 87–120. 2002.

[11] W. Ahn, W. F. Brewer, and R. J. Mooney. Schema acquisition from a single example. *Journal of Experimental Psychology: Learning, Memory and Cognition*, 18(2):391–412, 1992.

[12] A. Gelman, J. B. Carlin, H. S. Stern, and D. B. Rubin. *Bayesian data analysis*. Chapman & Hall, New York, 2nd edition, 2003.

[13] C. Kemp, N. D. Goodman, and J. B. Tenenbaum. Learning and using relational theories. In J.C. Platt, D. Koller, Y. Singer, and S. Roweis, editors, *Advances in Neural Information Processing Systems 20*, pages 753–760. MIT Press, Cambridge, MA, 2008.

[14] S. Kok and P. Domingos. Learning the structure of Markov logic networks. In *Proceedings of the 22nd International Conference on Machine Learning*, 2005.

[15] J. Feldman. The structure of perceptual categories. *Journal of Mathematical Psychology*, 41: 145–170, 1997.

[16] A. Jern and C. Kemp. Category generation. In *Proceedings of the 31st Annual Conference of the Cognitive Science Society*, pages 130–135. Cognitive Science Society, Austin, TX, 2009.

[17] D. Conklin and I. H. Witten. Complexity-based induction. *Machine Learning*, 16(3):203–225, 1994.

[18] J. B. Tenenbaum and T. L. Griffiths. Generalization, similarity, and Bayesian inference. *Behavioral and Brain Sciences*, 24:629–641, 2001.

[19] C. Kemp, A. Bernstein, and J. B. Tenenbaum. A generative theory of similarity. In B. G. Bara, L. Barsalou, and M. Bucciarelli, editors, *Proceedings of the 27th Annual Conference of the Cognitive Science Society*, pages 1132–1137. Lawrence Erlbaum Associates, 2005.

[20] C. Kemp, N. D. Goodman, and J. B. Tenenbaum. Theory acquisition and the language of thought. In *Proceedings of the 30th Annual Conference of the Cognitive Science Society*, pages 1606–1611. Cognitive Science Society, Austin, TX, 2008.

[21] K. J. Holyoak and P. Thagard. Analogical mapping by constraint satisfaction. *Cognitive Science*, 13(3):295–355, 1989.

[22] L. A. A. Doumas, J. E. Hummel, and C. M. Sandhofer. A theory of the discovery and predication of relational concepts. *Psychological Review*, 115(1):1–43, 2008.

[23] M. L. Gick and K. J. Holyoak. Schema induction and analogical transfer. *Cognitive Psychology*, 15:1–38, 1983.

